# Segmenting Scenes by Matching Image Composites

**Bryan C. Russell**[1] **Alexei A. Efros**[2,1] **Josef Sivic**[1] **William T. Freeman**[3] **Andrew Zisserman**[4,1]

[1]**INRIA**[*]    [2]**Carnegie Mellon University**    [3]**CSAIL MIT**    [4]**University of Oxford**

## Abstract

In this paper, we investigate how, given an image, similar images sharing the same global description can help with unsupervised scene segmentation. In contrast to recent work in semantic alignment of scenes, we allow an input image to be explained by partial matches of similar scenes. This allows for a better explanation of the input scenes. We perform MRF-based segmentation that optimizes over matches, while respecting boundary information. The recovered segments are then used to re-query a large database of images to retrieve better matches for the target regions. We show improved performance in detecting the principal occluding and contact boundaries for the scene over previous methods on data gathered from the LabelMe database.

## 1   Introduction

Segmenting semantic objects, and more broadly image parsing, is a fundamentally challenging problem. The task is painfully under-constrained – given a *single* image, it is extremely difficult to partition it into semantically meaningful elements, not just blobs of similar color or texture. For example, how would the algorithm figure out that doors and windows on a building, which look quite different, belong to the same segment? Or that the grey pavement and a grey house next to it are different segments? Clearly, information beyond the image itself is required to solve this problem.

In this paper, we argue that some of this extra information can be extracted by also considering images that are *visually similar* to the given one. With the increasing availability of Internet-scale image collections (in the millions of images!), this idea of data-driven scene matching has recently shown much promise for a variety of tasks. Simply by finding matching images using a low-dimential descriptor and transfering any associated labels onto the input image, impressive results have been demonstrated for object and scene recognition [22], object detection [18, 11], image geo-location [7], and particular object and event annotation [15], among others. Even if the image collection does not contain any labels, it has been shown to help tasks such as image completion and exploration [6, 21], image colorization [22], and 3D surface layout estimation [5].

However, as noted by several authors and illustrated in Figure 1, the major stumbling block of all the scene-matching approaches is that, despite the large quantities of data, for many types of images the quality of the matches is still not very good. Part of the reason is that the low-level image descriptors used for matching are just not powerful enough to capture some of the more semantic similarity. Several approaches have been proposed to address this shortcoming, including synthetically increasing the dataset with transformed copies of images [22], cleaning matching results using clustering [18, 7, 5], automatically prefiltering the dataset [21], or simply picking good matches by hand [6]. All these appraoches improve performance somewhat but don't alleviate this issue entirely. We believe that there is a more fundamental problem – the variability of the visual world is just so vast, with exponential number of different object combinations within each scene, that it might be

---

[*]WILLOW project-team, Laboratoire d'Informatique de l'École Normale Supérieure ENS/INRIA/CNRS UMR 8548

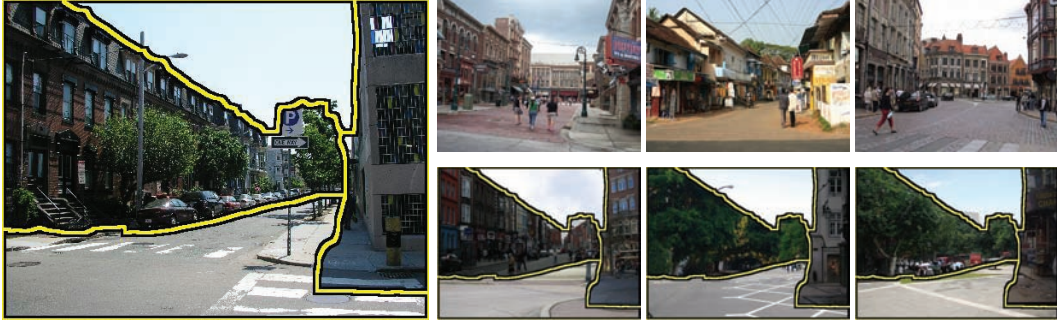

Figure 1: Illustration of the scene matching problem. Left: Input image (along with the output segmentation given by our system overlaid) to be matched to a dataset of 100k street images. Notice that the output segment boundaries align well with the depicted objects in the scene. Top right: top three retrieved images, based on matching the gist descriptor [14] over the entire image. The matches are not good. Bottom right: Searching for matches within each estimated segment (using the same gist representation within the segment) and compositing the results yields much better matches to the input image.

futile to expect to always find a single overall good match at all! Instead, we argue that an input image should be explained by a *spatial composite* of different regions taken from different database images. The aim is to break-up the image into chunks that are small enough to have good matches within the database, but still large enough that the matches retain their informative power.

## 1.1 Overview

In this work, we propose to apply scene matching to the problem of segmenting out semantically meaningful objects (i.e. we seek to segment objects enclosed by the principal occlusion and contact boundaries and not objects that are part-of or attached to other objects). The idea is to turn to our advantage the fact that scene matches are never perfect. What typically happens during scene matching is that some part of the image is matched quite well, while other parts are matched only approximately, at a very coarse level. For example, for a street scene, one matching image could have a building match very well, but getting the shape of the road wrong, while another matching image could get the road exactly right, but have a tree instead of a building. These differences in matching provide a powerful signal to identify objects and segmentation boundaries. By computing a matching image composite, we should be able to better explain the input image (i.e. match each region in the input image to semantically similar regions in other images) than if we used a single best match.

The starting point of our algorithm is an input image and an "image stack" – a set of coarsely matching images (5000 in our case) retrieved from a large dataset using a standard image matching technique (gist [14] in our case). In essence, the image stack is itself a dataset, but tailor-made to match the overall scene structure for the particular input image. Intuitively, our goal is to use the image stack to segment (and "explain") the input image in a semantically meaningful way. The idea is that, since the stack is already more-or-less aligned, the regions corresponding to the semantic objects that are present in many images will consistently appear in the same spatial location. The input image can then be explained as a patch-work of these consistent regions, simultaneously producing a segmentation, as well as composite matches, that are better than any of the individual matches within the stack.

There has been prior work on producing a resulting image using a stack of aligned images depicting the same scene, in particular the PhotoMontage work [1], which optimally selects regions from the globally aligned images based on a quality score to composite a visually pleasing output image. Recently, there has been work based on the PhotoMontage framework that tries to automatically align images depicting the same scene or objects to perform segmentation [16], region-filling [23], and outlier detection [10]. In contrast, in this work, we are attempting to work on a stack of visually similar, but physically different, scenes. This is in the same spirit as the contemporary work of [11],

except they work on supervised data, whereas we are completely unsupervised. Also related is the contemporary work of [9].

Our approach combines boundary-based and region-based segmentation processes together within a single MRF framework. The boundary process (Section 2) uses the stack to determine the likely semantic boundaries between objects. The region process (Section 3) aims to group pixels belonging to the same object across the stack. These cues are combined together within an MRF framework which is solved using GraphCut optimization (Section 4). We present results in Section 5.

## 2  Boundary process: data driven boundary detection

Information from only a single image is in many cases not sufficient for recovering boundaries between objects. Strong image edges could correspond to internal object structures, such as a window or a wheel of a car. Additionally, boundaries between objects often produce weak image evidence, as for example the boundary between a building and road of similar color partially occluding each other.

Here, we propose to analyze the statistics of a large number of related images (the stack) to help recover boundaries between objects. We will exploit the fact that objects tend not to rest at exactly the same location relative to each other in a scene. For example, in a street scene, a car may be adjacent to regions belonging to a number of objects, such as building, person, road, etc. On the other hand, relative positions of internal object structures will be consistent across many images. For example, wheels and windows on a car will appear consistently at roughly similar positions across many images.

To recover object boundaries, we will measure the ability to consistently match locally to the same set of images in the stack. Intuitively, regions inside an object will tend to match to the same set of images, each having similar appearance, while regions on opposite sides of a boundary will match to different sets of images. More formally, given an oriented line passing through an image point $p$ at orientation $\theta$, we wish to analyze the statistics of two sets of images with similar appearance on each side of the line. For each side of the oriented line, we independently query the stack of images by forming a local image descriptor modulated by a weighted mask. We use a half-Gaussian weighting mask oriented along the line and centered at image point $p$. This local mask modulates the Gabor filter responses (8 orientations over 4 scales) and the RGB color channels, with a descriptor formed by averaging the Gabor energy and color over $32 \times 32$ pixel spatial bins. The Gaussian modulated descriptor $\mathbf{g}(p, \theta)$ captures the appearance information on one side of the boundary at point $p$ and orientation $\theta$. Appearance descriptors extracted in the same manner across the image stack are compared with the query image descriptor using the L1 distance. Images in the stack are assumed to be coarsely aligned, and hence matches are considered only at the particular query location $p$ and orientation $\theta$ across the stack, i.e. matching is *not* translation invariant. We believe this type of spatially dependent matching is suitable for scene images with consistent spatial layout considered in this work. The quality of the matches can be further improved by fine aligning the stack images with the query [12].

For each image point $p$ and orientation $\theta$, the output of the local matching on the two sides of the oriented line are two ranked lists of image stack indices, $S_r$ and $S_l$, where the ordering of each list is given by the L1 distance between the local descriptors $\mathbf{g}(p, \theta)$ of the query image and each image in the stack. We compute Spearman's rank correlation coefficient between the two rank-ordered lists

$$\rho(p, \theta) = 1 - \frac{6\sum_{i=1}^{n} d_i^2}{n(n^2 - 1)}, \tag{1}$$

where $n$ is the number of images in the stack and $d_i$ is the difference between ranks of the stack image $i$ in the two ranked lists, $S_r$ and $S_l$. A high rank correlation should indicate that point $p$ lies inside an object's extent, whereas a low correlation should indicate that point $p$ is at an object boundary with orientation $\theta$. We note however, that low rank correlations could be also caused by poor quality of local matches. Figure 2 illustrates the boundary detection process.

For efficiency reasons, we only compute the rank correlation score along points and orientations marked as boundaries by the probability of boundary edge detector (PB) [13], with boundary orientations $\theta \in [0, \pi)$ quantized in steps of $\pi/8$. The final boundary score $P_{DB}$ of the proposed data

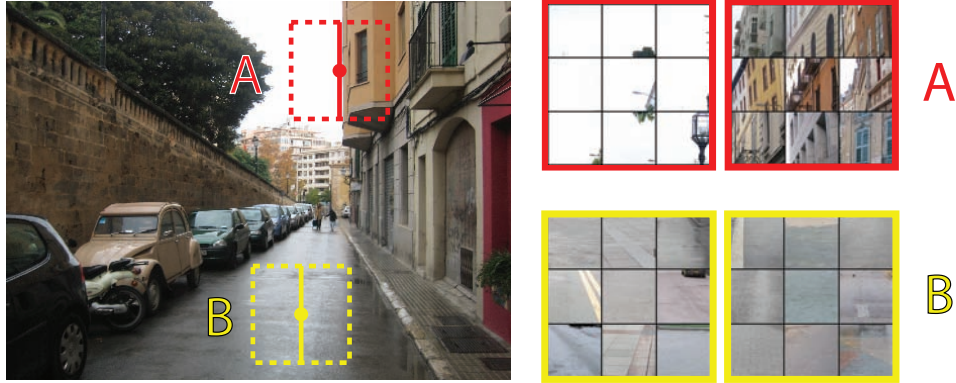

Figure 2: Data driven boundary detection. Left: Input image with query edges shown. Right: The top 9 matches in a large collection of images for each side of the query edges. Rank correlation for occlusion boundary (A): -0.0998; rank correlation within the road region (B): 0.6067. Notice that for point B lying inside an object (the road), the ranked sets of retrieved images for the two sides of the oriented line are similar, resulting in a high rank correlation score. At point A lying at an occlusion boundary between the building and the sky, the sets of retrieved images are very different, resulting in a low rank correlation score.

driven boundary detector is a gating of the maximum PB response over all orientations, $P_B$, and the rank correlation coefficient $\rho$,

$$P_{DB}(p, \theta) = P_B(p, \theta)\frac{1 - \rho(p, \theta)}{2}\delta[P_B(p, \theta) = \max_{\bar{\theta}} P_B(p, \bar{\theta})]. \tag{2}$$

Note that this type of data driven boundary detection is very different from image based edge detection [4, 13] as (i) strong image edges can receive a low score provided the matched image structures on each side of the boundary co-occur in many places in the image collection, and (ii) weak image edges can receive a high score, provided the neighboring image structures on each side of the weak image boundary do not co-occur often in the database. In contrast to the PB detector, which is trained from manually labelled object boundaries, data driven boundary scores are determined based on co-occurrence statistics of similar scenes and require no additional manual supervision. Figure 3 shows examples of data driven boundary detection results. Quantitative evaluation is given in section 5.

## 3 Region process: data driven image grouping

The goal is to group pixels in a query image that are likely to belong to the same object or a major scene element (such as a building, a tree, or a road). Instead of relying on local appearance similarity, such as color or texture, we again turn to the dataset of scenes in the image stack to suggest the groupings.

Our hypothesis is that regions corresponding to semantically meaningful objects would be coherent across a large part of the stack. Therefore, our goal is to find clusters within the stack that are both (i) self-consistent, and (ii) explain well the query image. Note that for now, we do not want to make any hard decisions, therefore, we want to allow multiple clusters to be able to explain overlapping parts of the query image. For example, a tree cluster and a building cluster (drawn from different parts of the stack) might be able to explain the same patch of the image, and both hypotheses should be retained. This way, the final segmentation step in the next section will be free to chose the best set of clusters based on all the information available within a global framework.

Therefore our approach is to find clusters of image patches that match the same images within the stack. In other words, two patches in the query image will belong to the same group if the sets of their best matching images from the database are similar. As in the boundary process described in section 2, the query image is compared with each database image only at the particular query patch location, i.e. the matching is not translation invariant. Note that patches with very different appearance can be grouped together as long as they match the same database images. For example, a

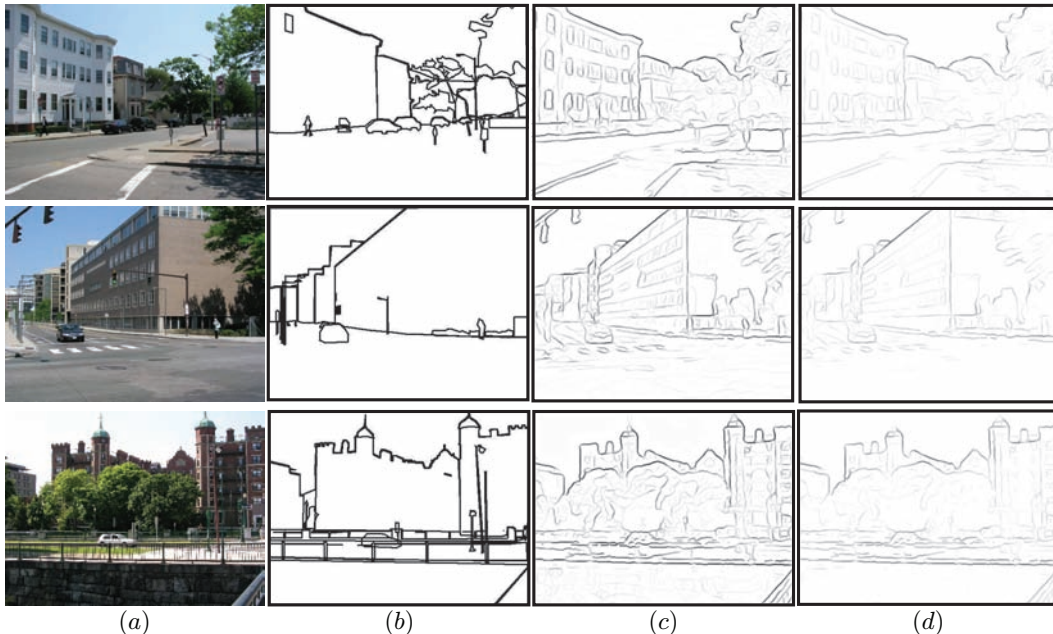

| (a) | (b) | (c) | (d) |

Figure 3: Data driven boundary detection. (a) Input image. (b) Ground truth boundaries. (c) $P_B$ [13]. (d) Proposed data driven boundary detection. Notice enhanced object boundaries and suppressed false positives boundaries inside objects.

door and a window of a building can be grouped together despite their different shape and appearance as long as they co-occur together (and get matched) in other images. This type of matching is different from self-similarity matching [20] where image patches within the same image are grouped together if they look similar.

Formally, given a database of $N$ scene images, each rectangular patch in the query image is described by an $N$ dimensional binary vector, $\mathbf{y}$, where the i-th element $y_{[i]}$ is set to 1 if the i-th image in the database is among the $m = 1000$ nearest neighbors of the patch. Other elements of $\mathbf{y}$ are set to 0. The nearest neighbors for each patch are obtained by matching the local gist and color descriptors at the particular image location as described in section 2, but here center weighted by a full Gaussian mask with $\sigma = 24$ pixels.

We now wish to find cluster centers $\mathbf{c}_k$ for $k \in \{1, \dots, K\}$. Many methods exist for finding clusters in such space. For example, one can think of the desired object clusters as "topics of an image stack" and apply one of the standard topic discovery methods like probabilistic latent semantic analysis (pLSA) [8] or Latent Dirichlet Allocation (LDA) [2]. However, we found that a simple K-means algorithm applied to the indicator vectors produced good results. Clearly, the number of clusters, $K$, is an important parameter. Because we are not trying to discover all the semantic objects within a stack, but only those that explain well the query image, we found that a relatively small number of clusters (e.g. 5) is sufficient. Figure 4 shows heat maps of the similarity (measured as $\mathbf{c}_k^T \mathbf{y}$) of each binary vector to the recovered cluster centers. Notice that regions belonging to the major scene components are highlighted. Although hard K-means clustering is applied to cluster patches at this stage, a soft similarity score for each patch under each cluster is used in a segmentation cost function incorporating both region and boundary cues, described next.

# 4 Image segmentation combining boundary and region cues

In the preceding two sections we have developed models for estimating data-driven scene boundaries and coherent regions from the image stack. Note that while both the boundary and the region processes use the same data, they are in fact producing very different, and complementary, types of information. The region process aims to find large groups of coherent pixels that co-occur together often, but is not too concerned about precise localization. The boundary process, on the other hand, focuses rather myopically on the local image behavior around boundaries but has excellent localiza-

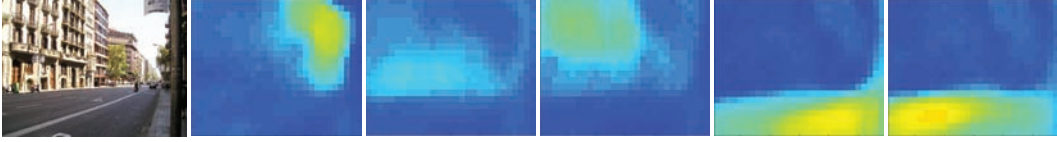

Figure 4: Data driven image grouping. Left: input image. Right: heat maps indicating groupings of pixels belonging to the same scene component, which are found by clustering image patches that match the same set of images in the stack (warmer colors correspond to higher similarity to a cluster center). Notice that regions belonging to the major scene components are highlighted. Also, local regions with different appearances (e.g. doors and windows in the interior of the building) can map to the same cluster since they only need to match to the same set of images. Finally, the highlighted regions tend to overlap, thereby providing multiple hypotheses for a local region.

tion. Both pieces of information are needed for a successful scene segmentation and explanation. In this section, we propose to use a single MRF-based optimization framework for this task, that will negotiate between the more global region process and the well-localized boundary process. We set up a multi-state MRF on pixels for segmentation, where the states correspond to the $K$ different image stack groups from section 3. The MRF is formulated as follows:

$$\min_{\mathbf{x}} \sum_i \phi_i(x_i, \mathbf{y}_i) + \sum_{(i,j)} \psi_{i,j}(x_i, x_j) \tag{3}$$

where $x_i \in \{0, 1, \ldots, K\}$ is the state at pixel $i$ corresponding to one of $K$ different image stack groups (section 3), $\phi_i$ are unary costs defined by similarity of a patch at pixel $i$, described by an indicator vector $\mathbf{y}_i$ (section 3), to each of the $K$ image stack groups, and $\psi_{i,j}$ are binary costs for a boundary-dependent Potts model (section 2). We also allow an additional *outlier* state $x_i = 0$ for regions that do not match any of the clusters well. For the pairwise term, we assume a 4-neighbourhood structure, i.e. the extent is over adjacent horizontal and vertical neighbors. The unary term in Equation 3 encourages pixels explained well by the same group of images from the stack to receive the same label. The binary term encourages neighboring pixels to have the same label, except in a case of a strong boundary evidence.

In more details, the unary term is given by

$$\phi_i(x_i = k, \mathbf{y}_i) = \begin{cases} -s(\mathbf{c}_k, \mathbf{y}_i) & k \in \{1, \ldots, K\} \\ \gamma & k = 0 \end{cases} \tag{4}$$

where $\gamma$ is a scalar parameter, and $s(\mathbf{c}_k, \mathbf{y}_i) = \mathbf{c}_k^T \mathbf{y}_i$ is the similarity between indicator vector $\mathbf{y}_i$ describing the local image appearance at pixel $i$ (section 3) and the k-th cluster center $\mathbf{c}_k$.

The pairwise term is defined as

$$\psi_{i,j}(x_i, x_j) = (\alpha + \beta f(i,j)) \, \delta[x_i = x_j] \tag{5}$$

where $f(i,j)$ is a function dependent on the output of the data-driven boundary detector $P_{DB}$ (Equation 2), and $\alpha$ and $\beta$ are scalar parameters. Since $P_{DB}$ is a line process with output strength and orientation defined at pixels rather than between pixels, as in the standard contrast dependent pairwise term [3], we must take care to place the pairwise costs consistently along one side of each continuous boundary. For this, let $P_i = \max_\theta P_{DB}(i, \theta)$ and $\theta_i = \mathrm{argmax}_\theta P_{DB}(i, \theta)$. If $i$ and $j$ are vertical neighbors, with $i$ on top, then $f(i,j) = \max\{0, P_j - P_i\}$. If $i$ and $j$ are horizontal neighbors, with $i$ on the left, then $f(i,j) = \max\{0, (P_j - P_i)\delta[\theta_j < \pi/2], (P_i - P_j)\delta[\theta_i \geq \pi/2]\}$. Notice that since $P_{DB}$ is non-negative everywhere, we only incorporate a cost into the model when the difference between adjacent $P_{DB}$ elements is positive.

We minimize Equation (3) using graph cuts with alpha-beta swaps [3]. We optimized the parameters on a validation set by manual tuning on the boundary detection task (section 5). We set $\alpha = -0.1$, $\beta = 0.25$, and $\gamma = -0.25$. Note that the number of recovered segments is not necessarily equal to the number of image stack groups $K$.

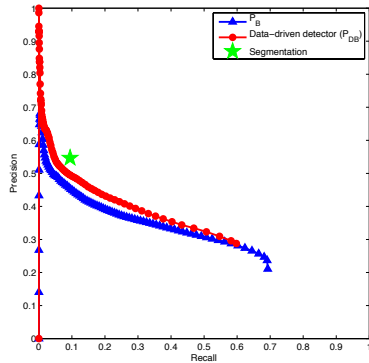

Figure 5: Evaluation of the boundary detection task on the principal occlusion and contact boundaries extracted from the LabelMe database [17]. We show precision-recall curves for $P_B$ [13] (blue triangle line) and our data-driven boundary detector (red circle line). Notice that we achieve improved performance across all recalls. We also show the precision and recall of the output segmentations (green star), which achieves 0.55 precision at 0.09 recall. At the same recall level, $P_B$ and the data-driven boundary detector achieves 0.45 and 0.50 precision, respectively.

## 5 Experimental evaluation

In this section, we evaluate the data-driven boundary detector and the proposed image segmentation model on a challenging dataset of complex street scenes from the LabelMe database [19]. For the unlabelled scene database, we use a dataset of 100k street scene images gathered from Flickr [21]. Boundary detection and image grouping are then applied only within this candidate set of images.

Figure 6 shows several final segmentations. Notice that the recovered segments correspond to the large objects depicted in the images, with the segment boundaries aligning along the objects' boundaries. For each segment, we re-query the image stack by using the segment as a weighted mask to retrieve images that match the appearance within the segment. The top matches for each segment are stitched together to form a composite, which are shown in Figure 6. As a comparison, we show the top matches using the global descriptor. Notice that the composites better align with the contents depicted in the input image.

We quantitatively evaulate our system by measuring how well we can detect ground truth object boundaries provided by human labelers. To evaluate object boundary detection, we use 100 images depicting street scenes from the benchmark set of the LabelMe database [19]. The benchmark set consists of fully labeled images taken from around the world. A number of different types of edges are implicitly labeled in the LabelMe database, such as those arising through occlusion, attachment, and contact with the ground. For this work, we filter out attached objects (e.g. a window is attached to a building and hence does not generate any object boundaries) using the techniques outlined in [17]. Note that this benchmark is more appropriate for our task than the BSDS [13] since the dataset explicitly contains occlusion boundaries and not interior contours.

To measure performance, we used the evaluation procedure outlined in [13], which aligns output boundaries for a given threshold to the ground truth boundaries to compute precision and recall. A curve is generated by evaluating at all thresholds. For a boundary to be considered correct, we assume that it must lie within 6 pixels of the ground truth boundary.

Figure 5 shows a precision-recall curve for the data-driven boundary detector. We compare against PB using color [13]. Notice that we achieve higher precision at all recall levels. We also plot the precision and recall of the output segmentation produced by our system. Notice that the segmentation produced the highest precision (0.55) at 0.09 recall. The improvement in performance at low recall is largely due to the ability to suppress interior contours due to attached objects (c.f. Figure 3). However, we tend to miss small, moveable objects, which accounts for the lower performance at high recall.

## 6 Conclusion

We have shown that unsupervised analysis of a large image collection can help segmenting complex scenes into semantically coherent parts. We exploit object variations over related images using MRF-based segmentation that optimizes over matches while preserving scene boundaries obtained by a data driven boundary detection process. We have demonstrated an improved performance in detecting the principal occlusion and contact boundaries over previous methods on a challenging dataset of complex street scenes from LabelMe. Our work also suggests that other applications of

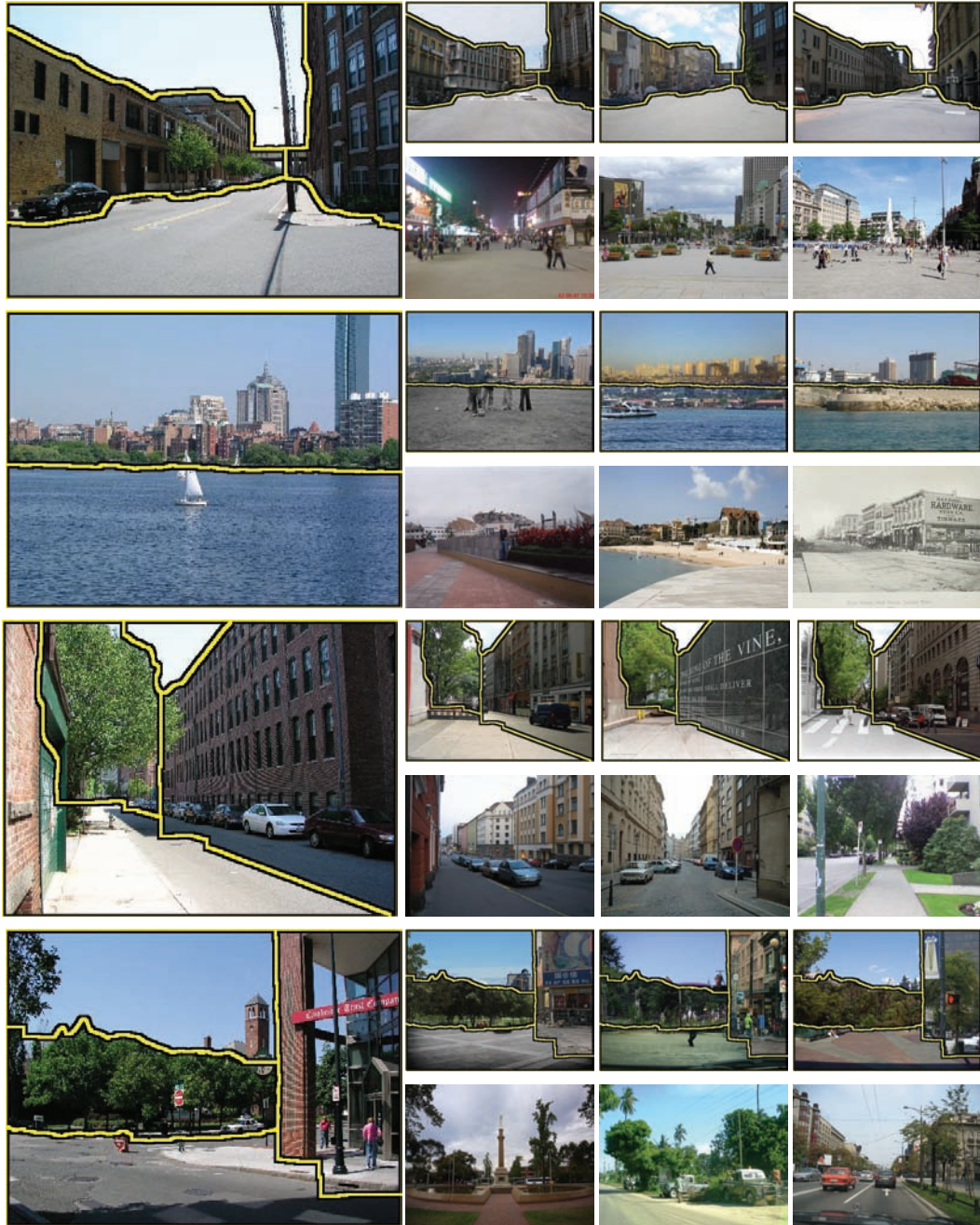

Figure 6: Left: Output segmentation produced by our system. Notice that the segment boundaries align well with the depicted objects in the scene. Top right: Top matches for each recovered segment, which are stitched together to form a composite. Bottom right: Top whole-image matches using the gist descriptor. By recovering the segmentation, we are able to recover improved semantic matches.

scene matching, such as object recognition or computer graphics, might benefit from segment-based explanations of the query scene.

**Acknowledgments:** This work was partially supported by ONR MURI N00014-06-1-0734, ONR MURI N00014-07-1-0182, NGA NEGI-1582-04-0004, NSF grant IIS-0546547, gifts from Microsoft Research and Google, and Guggenheim and Sloan fellowships.

# References

[1] A. Agarwala, M. Dontcheva, M. Agrawala, S. Drucker, A. Colburn, B. Curless, D. Salesin, and M. Cohen. Interactive digital photomontage. In *SIGGRAPH*, 2004.

[2] D. Blei, A. Ng, and M. Jordan. Latent dirichlet allocation. *Journal of Machine Learning Research*, 3:993–1022, 2003.

[3] Y. Boykov, O. Veksler, and R. Zabih. Fast approximate energy minimization via graph cuts. *IEEE Trans. on Pattern Analysis and Machine Intelligence*, 23(11), 2001.

[4] J. F. Canny. A computational approach to edge detection. *IEEE Trans. on Pattern Analysis and Machine Intelligence*, 8(6):679–698, 1986.

[5] S. K. Divvala, A. A. Efros, and M. Hebert. Can similar scenes help surface layout estimation? In *IEEE Workshop on Internet Vision, associated with CVPR*, 2008.

[6] J. Hays and A. Efros. Scene completion using millions of photographs. In *"SIGGRAPH"*, 2007.

[7] J. Hays and A. A. Efros. IM2GPS: estimating geographic information from a single image. In *CVPR*, 2008.

[8] T. Hofmann. Unsupervised learning by probabilistic latent semantic analysis. *Machine Learning*, 43:177–196, 2001.

[9] M. K. Johnson, K. Dale, S. Avidan, H. Pfister, W. T. Freeman, and W. Matusik. CG2Real: Improving the realism of computer-generated images using a large collection of photographs. Technical Report 2009-034, MIT CSAIL, 2009.

[10] H. Kang, A. A. Efros, M. Hebert, and T. Kanade. Image composition for object pop-out. In *IEEE Workshop on 3D Representation for Recognition (3dRR-09), in assoc. with CVPR*, 2009.

[11] C. Liu, J. Yuen, and A. Torralba. Nonparametric scene parsing: label transfer via dense scene alignment. In *CVPR*, 2009.

[12] C. Liu, J. Yuen, A. Torralba, J. Sivic, and W. T. Freeman. SIFT flow: dense correspondence across different scenes. In *ECCV*, 2008.

[13] D. Martin, C. Fowlkes, and J. Malik. Learning to detect natural image boundaries using local brightness, color, and texture cues. *IEEE Trans. on Pattern Analysis and Machine Intelligence*, 26(5):530–549, 2004.

[14] A. Oliva and A. Torralba. Modeling the shape of the scene: a holistic representation of the spatial envelope. *IJCV*, 42(3):145–175, 2001.

[15] T. Quack, B. Leibe, and L. V. Gool. World-scale mining of objects and events from community photo collections. In *CIVR*, 2008.

[16] C. Rother, V. Kolmogorov, T. Minka, and A. Blake. Cosegmentation of image pairs by histogram matching - incorporating a global constraint into MRFs. In *CVPR*, 2006.

[17] B. C. Russell and A. Torralba. Building a database of 3D scenes from user annotations. In *CVPR*, 2009.

[18] B. C. Russell, A. Torralba, C. Liu, R. Fergus, and W. T. Freeman. Object recognition by scene alignment. In *Advances in Neural Info. Proc. Systems*, 2007.

[19] B. C. Russell, A. Torralba, K. P. Murphy, and W. T. Freeman. LabelMe: a database and web-based tool for image annotation. *IJCV*, 77(1-3):157–173, 2008.

[20] E. Shechtman and M. Irani. Matching local self-similarities across images and videos. In *CVPR*, 2007.

[21] J. Sivic, B. Kaneva, A. Torralba, S. Avidan, and W. T. Freeman. Creating and exploring a large photorealistic virtual space. In *First IEEE Workshop on Internet Vision, associated with CVPR*, 2008.

[22] A. Torralba, R. Fergus, and W. T. Freeman. 80 million tiny images: a large dataset for non-parametric object and scene recognition. *IEEE Trans. on Pattern Analysis and Machine Intelligence*, 30(11):1958–1970, 2008.

[23] O. Whyte, J. Sivic, and A. Zisserman. Get out of my picture! Internet-based inpainting. In *British Machine Vision Conference*, 2009.

